# Group Anomaly Detection using Flexible Genre Models

**Liang Xiong**
Machine Learning Department,
Carnegie Mellon University
lxiong@cs.cmu.edu

**Barnabás Póczos**
Robotics Institute,
Carnegie Mellon University
bapoczos@cs.cmu.edu

**Jeff Schneider**
Robotics Institute,
Carnegie Mellon University
schneide@cs.cmu.edu

## Abstract

An important task in exploring and analyzing real-world data sets is to detect unusual and interesting phenomena. In this paper, we study the group anomaly detection problem. Unlike traditional anomaly detection research that focuses on data points, our goal is to discover anomalous aggregated behaviors of groups of points. For this purpose, we propose the *Flexible Genre Model* (FGM). FGM is designed to characterize data groups at both the point level and the group level so as to detect various types of group anomalies. We evaluate the effectiveness of FGM on both synthetic and real data sets including images and turbulence data, and show that it is superior to existing approaches in detecting group anomalies.

## 1   Introduction

Anomaly detection is a crucial problem in processing large-scale data sets when our goal is to find rare or unusual events. These events can either be outliers that should be ignored or novel observations that could lead to new discoveries. See [1] for a recent survey of this field. Traditional research often focuses on individual data points. In this paper, however, we are interested in finding *group anomalies*, where a set of points together exhibit unusual behavior. For example, consider text data where each article is considered to be a set (group) of words (points). While the phrases "machine learning" or "gummy bears" will not surprise anyone on their own, an article containing both of them might be interesting.

We consider two types of group anomalies. A *point-based* group anomaly is a group of individually anomalous points. A *distribution-based* anomaly is a group where the points are relatively normal, but as a whole they are unusual. Most existing work on group anomaly detection focuses on point-based anomalies. A common way to detect point-based anomalies is to first identify anomalous points and then find their aggregations using scanning or segmentation methods [2, 3, 4]. This paradigm clearly does not work well for distribution-based anomalies, where the individual points are normal. To handle distribution-based anomalies, we can design features for groups and then treat them as points [5, 6]. However, this approach relies on feature engineering that is domain specific and can be difficult. Our contribution is to propose a new method (FGM) for detecting both types of group anomalies in an integral way.

Group anomalies exist in many real-world problems. In astronomical studies, modern telescope pipelines[1] produce descriptions for a vast amount of celestial objects. Having these data, we want to pick out scientifically valuable objects like planetary nebulae, or special clusters of galaxies that could shed light on the development of the universe [7]. In physics, researchers often simulate the motion of particles or fluid. In these systems, a single particle is seldom interesting, but a group of particles can exhibit interesting motion patterns like the interweaving of vortices. Other examples are abundant in the fields of computer vision, text processing, time series and spatial data analysis.

We take a generative approach to address this problem. If we have a model to generate normal data, then we can mark the groups that have small probabilities under this model as anomalies. Here we make the "bag-of-points" assumption, *i.e.*, points in the same group are unordered and *exchangeable*. Under this assumption, mixture models are often used to generate the data due to *De Finetti*'s theorem [8]. The most famous class of mixture models for modeling group data is the family of *topic models* [9, 10]. In topic models, distributions of points in different groups are mixtures of components ("topics"), which are shared among all the groups.

Our proposed method is closely related to the class of topic models, but it is designed specifically for the purpose of detecting group anomalies. We use two levels of concepts/latent variables to describe a group. At the group level, a flexible structure based on "genres" is used to characterize the topic distributions so that complex normal behaviors are allowed and can be recognized. At the point level, each group has its own topics to accommodate and capture the variations of points' distributions (while global topic information is still shared among groups). We call this model the *Flexible Genre Model* (FGM). Given a group of points, we can examine whether or not it conforms to the normal behavior defined by the learned genres and topics. We will also propose scoring functions that can detect both point-based and distribution-based group anomalies. Exact inference and learning for FGM is intractable, so we resort to approximate methods. Inference for the FGM model will be done by *Gibbs sampling* [11], which is efficient and simple to implement due to the application of conjugate distributions. Single-sample Monte Carlo EM [12] is used to learn parameters based on samples produced by the Gibbs sampler.

We demonstrate the effectiveness of the FGM on synthetic and on real-world data sets including scene images and turbulence data. Empirical results show that FGM is superior to existing approaches in finding group anomalies.

The paper is structured as follows. In Section 2 we review related work and discuss the limitations with existing algorithms and why a new method is needed for group anomaly detection. Section 3 introduces our proposed model. The parameter learning of our model and inference on it are explained in Section 4. Section 5 describes how to use our method for group anomaly detection. Experimental results are shown in Section 6. We finish that paper by drawing conclusions (Section 7).

## 2   Background and Related Work

In this section, we provide background about topic models and explain the limitation of existing methods in detecting group anomalies. For intuition, we introduce the problem in the context of detecting anomalous images, rare galaxy clusters, and unusual motion in a dynamic fluid simulation.

We consider a data set with $M$ pre-defined groups $\mathbf{G}_1, \ldots, \mathbf{G}_M$ (e.g. spatial clusters of galaxies, patches in an image, or fluid motions in a local region). Group $\mathbf{G}_m$ contains $N_m$ points (galaxies, image patches, simulation grid points). The features of these points are denoted by $\mathbf{x}_m = \{x_{m,n} \in \mathbb{R}^f\}_{n=1,\ldots,N_m}$, where $f$ is the dimensionality of the points' features. These would be spectral features of each galaxy, SIFT features of each image patch, or velocities at each grid point of a simulation. We assume that points in the same group are unordered and exchangeable. Having these data, we ask the question whether in group $\mathbf{G}_m$ the distribution of features $\mathbf{x}_m$ looks anomalous.

Topic models such as *Latent Dirichlet Allocation* (LDA) [10] are widely used to model data having this kind of group structure. The original LDA model was proposed for text processing. It represents the distribution of points (words) in a group (document) as a mixture of $K$ global topics $\beta_1, \ldots \beta_K$, each of which is a distribution (*i.e.*, $\beta_i \in \mathbb{S}^f$, where $\mathbb{S}^f$ is the $f$-dimensional probability simplex). Let $\mathcal{M}(\theta)$ be the multinomial distribution parameterized by $\theta \in \mathbb{S}^K$ and $\mathcal{D}ir(\alpha)$ be the Dirichlet distribution with parameter $\alpha \in \mathbb{R}_+^K$. LDA generates the $m$th group by first drawing its topic distribution $\theta_m$ from the prior distribution $\mathcal{D}ir(\alpha)$. Then for each point $x_{mn}$ in the $m$th group it draws one of the $K$ topics from $\mathcal{M}(\theta_m)$ (*i.e.*, $z_{mn} \sim \mathcal{M}(\theta_m)$) and then generates the point according to this topic ($x_{mn} \sim \mathcal{M}(\beta_{z_{mn}})$).

In our examples, the topics can represent galaxy types (*e.g.* "blue","red", or "emissive", with $K = 3$), image features (e.g. edge detectors representing various orientations), or common motion patterns in the fluid (fast left, slow right, etc). Each point in the group has its own topic. We consider points that have multidimensional continuous feature vectors. In this case, topics can be

modeled by Gaussian distributions, and each point is generated from one of the $K$ Gaussian topics. At a higher level, a group is characterized by the distribution of topics $\theta_m$, *i.e.*, the proportion of different types in the group $\mathbf{G}_m$. The concepts of topic and topic distribution help us define group anomalies: a *point-based* anomaly contains points that do not belong to any of the normal topics and a *distribution-based* anomaly has a topic distribution $\theta_m$ that is uncommon.

Although topic models are very useful in estimating the topics and topic distributions in groups, the existing methods are *incapable of detecting group anomalies comprehensively*. In order to detect anomalies, the model should be flexible enough to enable complex normal behaviors. For example, it should be able to model complex and multi-modal distributions of the topic distribution $\theta$. LDA, however, only uses a single Dirichlet distribution to generate topic distributions, and cannot effectively define what normal and abnormal distributions should be. It also uses the same $K$ topics for every group, which makes groups indifferentiable when looking at their topics. In addition, these shared topics are not adapted to each group either.

The Mixture of Gaussian Mixture Model (MGMM) [13] firstly uses topic modeling for group anomaly detection. It allows groups to select their topic distributions from a dictionary of multinomials, which is learned from data to define what is normal. [14] employed the same idea but did not apply their model to anomaly detection. The problem of using multinomials is that it does not consider the uncertainty of topic distributions. The Theme Model (ThM) [15] lets a mixture of Dirichlets generate the topic distributions and then uses the memberships in this mixture to do clustering on groups. This idea is useful for modeling group-level behaviors but fails to capture anomalous point-level behaviors. The topics are still shared globally in the same way as in LDA. In contrast, [16] proposed to use different topics for different groups in order to account for the burstiness of the words (points). These adaptive topics are useful in recognizing point-level anomalies, but cannot be used to detect anomalous behavior at the group level. For the group anomaly detection problem we propose a new method, the *Flexible Genre Model*, and demonstrate that it is able to cope with the issues mentioned above and performs better than the existing state-of-the-art algorithms.

## 3 Model Specification

The flexible genre model (FGM) extends LDA such that the generating processes of topics and topic distributions can model more complex distributions. To achieve this goal, two key components are added. (i) To model the behavior of topic distributions, we use several "genres", each of which is a typical distribution of topic distributions. (ii) We use "topic generators" to generate adaptive topics for different groups. We will also use them to learn how the normal topics have been generated. The generative process of FGM is presented in Algorithm 1. A graphical representation of FGM is given in Figure 1.

---

**Algorithm 1** Generative process of FGM

   **for** Groups $m = 1$ to $M$ **do**
      • Draw a genre $\{1, \ldots, T\} \ni y_m \sim \mathcal{M}(\pi)$.
      • Draw a topic distribution according to the genre $y_m$: $\mathbb{S}^K \ni \theta_m \sim \mathcal{D}ir(\alpha_{y_m})$.
      • Draw $K$ topics $\{\beta_{m,k} \sim P(\beta_{m,k}|\eta_k)\}_{k=1,\ldots,K}$.
     **for** Points $n = 1$ to $N_m$ **do**
        • Draw a topic membership $\{1, \ldots, K\} \ni z_{m,n} \sim \mathcal{M}(\theta_m)$.    $[\beta_{m,z_{mn}}$ topic will be active.$]$
        • Generate a point $x_{m,n} \sim P(x_{m,n}|\beta_{m,z_{mn}})$.
     **end for**
   **end for**

---

We assume there are $T$ genres and $K$ topics. $\mathcal{M}(\pi)$ denotes the global distribution of genres. Each genre is a Dirichlet distribution for generating the topic distributions, and $\alpha = \{\alpha_t\}_{t=1,\ldots,T}$ is the set of genre parameters. Each group has $K$ topics $\beta_m = \{\beta_{m,k}\}_{k=1,\ldots,K}$. The "topic generators", $\eta = \{\eta_k\}, \{P(\cdot|\eta_k)\}_{k=1,\ldots,K}$, are the global distributions for generating the corresponding topics. Having the topic distribution $\theta_m$ and the topics $\{\beta_{m,k}\}$, points are generated as in LDA.

By comparing FGM to LDA, the advantages of FGM become evident. (i) In FGM, each group has a latent genre attribute $y_m$, which determines how the topic distribution in this group *should* look like ($Dir(\alpha_{y_m})$), and (ii) each group has its own topics $\{\beta_{m,k}\}_{k=1}^K$, but they are still tied through the

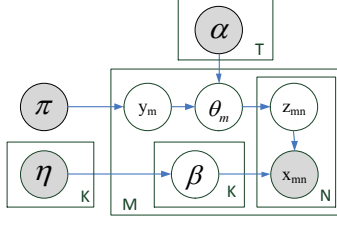

Figure 1: The Flexible Genre Model (FGM).

global distributions $P(\cdot|\eta)$. Thus, the topics can be adapted to local group data, but the information is still shared globally. Moreover, the topic generators $P(\cdot|\eta)$ determine how the topics $\{\beta_{m,k}\}$ *should* look like. In turn, if a group uses unusual topics to generate its points, it can be identified.

To handle real-valued multidimensional data, we set the point-generating distributions (*i.e.*, the topics) to be Gaussians, $P(x_{m,n}|\beta_{m,k}) = \mathcal{N}(x_{m,n}|\beta_{m,k})$, where $\beta_{m,k} = \{\mu_{m,k}, \Sigma_{m,k}\}$ includes the mean and covariance parameters. For computational convenience, the topic generators are *Gaussian-Inverse-Wishart* (GIW) distributions, which are conjugate to the Gaussian topics. Hence $\eta_k = \{\mu_0, \kappa_0, \Psi_0, \nu_0\}$ parameterizes the GIW distribution [17] (See the supplementary materials for more details). Let $\Theta = \{\pi, \alpha, \eta\}$ denote the model parameters. We can write the complete likelihood of data and latent variables in group $\mathbf{G}_m$ under FGM as follows:

$$P(\mathbf{G}_m, y_m, \theta_m, \beta_m|\Theta)$$
$$= \mathcal{M}(y_m|\pi)\mathcal{D}ir(\theta_m|\alpha_{y_m}) \prod_k GIW(\beta_{m,k}|\eta_k) \prod_n \mathcal{M}(z_{mn}|\theta_m)\mathcal{N}(x_{mn}|\beta_{m,z_{mn}}).$$

By integrating out $\theta_m, \beta_m$ and summing out $y_m, \mathbf{z}$, we get the marginal likelihood of $\mathbf{G}_m$:

$$P(\mathbf{G}_m|\Theta) = \sum_t \pi_t \int_{\theta_m,\beta_m} \mathcal{D}ir(\theta_m|\alpha_t) \prod_k GIW(\beta_{m,k}|\eta_k) \prod_n \sum_k \theta_{mk}\mathcal{N}(x_{mn}|\beta_{m,k})d\beta_m d\theta_m.$$

Finally, the data-set's likelihood is just the product of all groups' likelihoods.

## 4 Inference and Learning

To learn FGM, we update the parameters $\Theta$ to maximize the likelihood of data. The inferred latent states—including the topic distributions $\theta_m$, the topics $\beta_m$, and the topic and genre memberships $\mathbf{z}_m, y_m$—can be used for detecting anomalies and exploring the data. Nonetheless, the inference and learning in FGM is intractable, so we train FGM using an approximate method described below.

### 4.1 Inference

The approximate inference of the latent variables can be done using *Gibbs sampling* [11]. In Gibbs sampling, we iteratively update one variable at a time by drawing samples from its conditional distribution when all the other parameters are fixed. Thanks to the use of conjugate distributions, Gibbs sampling in FGM is simple and easy to implement. The sampling distributions of the latent variables in group $m$ are given below. We use $P(\cdot|\sim)$ to denote the distribution of one variable conditioned on all the others. For the genre membership $y_m$ we have that:

$$P(y_m = t|\sim) \propto P(\theta_m|\alpha_t)P(y_m = t|\pi) = \pi_t \mathcal{D}ir(\theta_m|\alpha_t).$$

For the topic distribution $\theta_m$:

$$P(\theta_m|\sim) \propto P(\mathbf{z}_m|\theta_m)P(\theta_m|\alpha, y_m) = \mathcal{M}(\mathbf{z}_m|\theta_m)\mathcal{D}ir(\theta_m|\alpha_{y_m}) = \mathcal{D}ir(\alpha_{y_m} + \mathbf{n}_m),$$

where $\mathbf{n}_m$ denotes the histogram of the $K$ values in vector $\mathbf{z}_m$. The last equation follows from the Dirichlet-Multinomial conjugacy. For $\beta_{m,k}$, the $k$th topic in group $m$, one can find that:

$$P(\beta_{m,k}|\sim) \propto P(\mathbf{x}_m^{(k)}|\beta_{m,k})P(\beta_{m,k}|\eta_k) = \mathcal{N}(\mathbf{x}_m^{(k)}|\beta_{m,k})GIW(\beta_{m,k}|\eta_k) = GIW(\beta_{m,k}|\eta'_k),$$

where $\mathbf{x}_m^{(k)}$ are points in group $\mathbf{G}_m$ from topic $k$, *i.e.*, $z_{m,n} = k$. The last equation follows from the Gaussian-Inverse-Wishart-Gaussian conjugacy. $\eta_k'$ is the parameter of the posterior GIW distribution given $x_m^{(k)}$; its exact form can be found in the supplementary material. For $z_{mn}$, the topic membership of point $n$ in group $m$ is as follows:

$$P(z_{mn} = k| \sim) \propto P(x_{mn}|z_{mn} = k, \beta_m)P(z_{mn} = k|\theta_m) = \theta_{m,k}\mathcal{N}(x_{mn}|\beta_{m,k}).$$

## 4.2 Learning

Learning the parameters of FGM helps us identify the groups' and points' normal behaviors. Each of the genres $\alpha = \{\alpha_t\}_{t=1,\ldots,T}$ captures one typical distribution of topic distributions as $\theta \sim \mathcal{D}ir(\alpha_t)$. The topic generators $\eta = \{\eta_k\}_{k=1,\ldots,K}$ determine how the normal topics $\{\beta_{m,k}\}$ should look like. We use single-sample Monte Carlo EM [12] to learn parameters from the samples provided by the Gibbs sampler. Given sampled latent variables, we update the parameters to their maximum likelihood estimations (MLE): we learn $\alpha$ from $\mathbf{y}$ and $\theta$; $\eta$ from $\beta$; and $\pi$ from $\mathbf{y}$.

$\pi$ can easily be estimated from the histogram of $y$'s. $\alpha_t$ is learned by the MLE of a Dirichlet distribution given the multinomials $\{\theta_m|y_m = t, m = 1,\ldots,M\}$ (*i.e.*, the topic distributions having genre $t$), which can be solved using the *Newton–Raphson* method [18]. The $k$th topic-generator's parameter $\eta_k = \{\mu_{0k}, \kappa_{0k}, \Psi_{0k}, \nu_{0k}\}$ is the MLE of a GIW distribution given the parameters $\{\beta_{m,k} = (\mu_{m,k}, \Sigma_{m,k})\}_{m=1,\ldots,M}$ (the $k$th topics of all groups). We have derived an efficient solution for this MLE problem. The details can be found in the supplementary material.

The overall learning algorithm works by repeating the following procedure until convergence: (1) do Gibbs sampling to infer the states of the latent variables; (2) update the model parameters using the estimations above. To select appropriate values for the parameters $T$ and $K$ (the number of genres and topics), we can apply the Bayesian information criterion (BIC) [19], or use the values that maximize the likelihood of a held-out validation set.

# 5 Scoring Criteria

The learned FGM model can easily be used for anomaly detection on test data. Given a test group, we first infer its latent variables including the topics and the topic distribution. Then we treat these latent states as the group's characteristicsand examine if they are compatible with the normal behaviors defined by the FGM parameters.

Point-based group anomalies can be detected by examining the topics of the groups. If a group contains anomalous points with rare feature values $x_{mn}$, then the topics $\{\beta_{m,k}\}_{k=1}^{K}$ that generate these points will deviate from the normal behavior defined by the topic generators $\eta$. Let $P(\beta_m|\Theta) = \prod_{k=1}^{K} GIW(\beta_{m,k}|\eta_k)$. The *point-based anomaly score* (PB score) of group $\mathbf{G}_m$ is

$$\mathbb{E}_{\beta_m}[-\ln P(\beta_m|\Theta)] = -\int_{\beta_m} P(\beta_m|\Theta, \mathbf{G}_m) \ln P(\beta_m|\Theta)\mathrm{d}\beta_m.$$

The posterior $P(\beta_m|\Theta, \mathbf{G}_m)$ can again be approximated using Gibbs sampling, and the expectation can be done by Monte Carlo integration.

Distribution-based group anomalies can be detected by examining the topic distributions. The genres $\{\alpha_t\}_{t=1,\ldots,T}$ capture the typical distributions of topic distributions. If a group's topic distribution $\theta_m$ is unlikely to be generated from any of these genres, we call it anomalous. Let $P(\theta_m|\Theta) = \sum_{t=1}^{T} \pi_t \mathcal{D}ir(\theta_m|\alpha_t)$. The *distribution-based anomaly score* (DB score) of group $\mathbf{G}_m$ is defined as

$$\mathbb{E}_{\theta_m}[-\ln P(\theta_m|\Theta)] = -\int_{\theta_m} P(\theta_m|\Theta, \mathbf{G}_m) \ln P(\theta_m|\Theta)\mathrm{d}\theta_m. \tag{1}$$

Again, this expectation can be approximated using Gibbs sampling and Monte Carlo integration. Using a combination of the point-based and distribution-based scores, we can detect both point-based and distribution-based group anomalies.

# 6 Experiments

In this section we provide empirical results produced by FGM on synthetic and real data. We show that FGM outperforms several sate-of-the-art competitors in the group anomaly detection task.

## 6.1 Synthetic Data

In the first experiment, we compare FGM with the Mixture of Gaussian Mixture Model (MGMM) [13] and with an adaptation of the Theme Model (ThM) [15] on synthetic data sets. The original ThM handles only discrete data and was proposed for clustering. To handle continuous data and detect anomalies, we modified it by using Gaussian topics and applied the distribution-based anomaly scoring function (1). To detect both distribution-based and point-based anomalies, we can use the data's likelihood under ThM as the scoring function.

Using the synthetic data sets described below, we can demonstrate the behavior of the different models and scoring functions. We generated the data using 2-dimensional GMMs as in [13]. Here each group has a GMM to generate its points. All GMMs share three Gaussian components with covariance $0.2 \times \mathbf{I}_2$ and centered at points $(-1.7, -1)$, $(1.7, -1)$, and $(0, 2)$, respectively. A group's mixing weights are randomly chosen from $w_1 = [0.33, 0.33, 0.33]$ or $w_2 = [0.84, 0.08, 0.08]$. Thus, a group is *normal* if its points are sampled from these three Gaussians, and their mixing weights are close to either $w_1$ or $w_2$. To test the detectors, we injected both point-based and distribution-based anomalies. Point-based anomalies were groups of points sampled from $\mathcal{N}((0, 0), \mathbf{I}_2)$. Distribution-based anomalies were generated by GMMs consisting of normal Gaussian components but with mixing weights $[0.33, 0.64, 0.03]$ and $[0.08, 0.84, 0.08]$, which were different from $w_1$ and $w_2$. We generated $M = 50$ groups, each of which had $N_m \sim Poisson(100)$ points. One point-based anomalous group and two distribution-based anomalous groups were injected into the data set.

The detection results of MGMM, ThM, and FGM are shown in Fig. 2. We show 12 out of the 50 groups. Normal groups are surrounded by black solid boxes, point-based anomalies have green dashed boxes, and distribution-based anomalies have red/magenta dashed boxes. Points are colored by the anomaly scores of the groups (darker color means more anomalous). An ideal detector would make dashed boxes' points dark and solid boxes' points light gray. We can see that all the

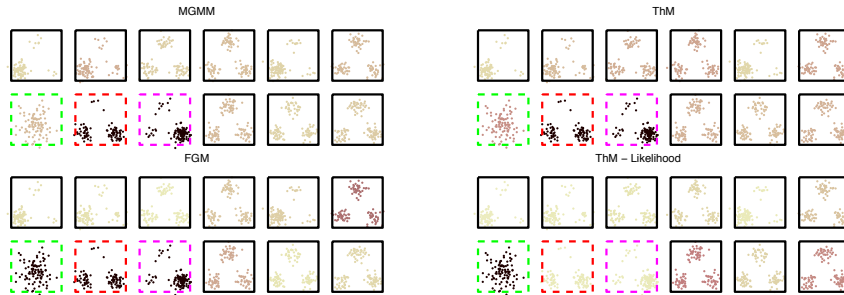

Figure 2: Detection results on synthetic data.

models can find the distribution-based anomalies since they are able to learn the topic distributions. However, MGMM and ThM miss the point-based anomaly. The explanation is simple; the anomalous points are distributed in the middle of the topics, thus the inferred topic distribution is around $[0.33, 0.33, 0.33]$, which is exactly $w_1$. As a result, MGMM and ThM infer this group to be normal, although it is not. This example shows one possible problem of scoring groups based on topic distributions only. On the contrary, using the sum of point-based and distribution-based scores, FGM found all of the group anomalies thanks to its ability to characterize groups both at the point-level and the group-level. We also show the result of scoring the groups by the ThM likelihood. Only point anomalies are found. This is because the data likelihood under ThM is dominated by the anomalousness of points, thus a few eccentric points will overshadow group-level behaviors.

Figures 3(a) – 3(c) show the density estimations given by MGMM, ThM, and FGM, respectively, for the point-based anomalous group. We can see that FGM gives a better estimation due to its adaptive topics, while MGMM and ThM are limited to use their global topics. Figure 3(d) shows the learned

genres visualized as the distribution $\sum_{t=1}^{T} \pi_t \mathcal{D}ir(\cdot|\alpha_t)$ on the topic simplex. This distribution summarizes the normal topic distributions in this data set. Observe that the two peaks in the probability simplex are very close to $w_1$ and $w_2$ indeed.

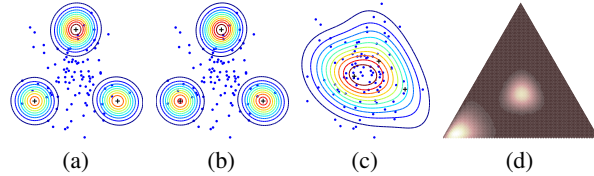

(a)        (b)        (c)        (d)

Figure 3: (a),(b),(c) show the density of the point-based anomaly estimated by MGMM, ThM, and FGM respectively. In MGMM and ThM, topics must be shared globally, therefore their perform badly. (d) The genres in the synthetic data set learned by FGM.

## 6.2   Image Data

In this experiment we test the performance of our method on detecting anomalous scene images. We use the data set from [15]. We selected the first 100 images from categories "mountain", "coast", and "inside city". These 300 images are randomly divided: $80\%$ are used for training and the rest for testing. We created anomalies by stitching random normal test images from different categories. For example, an anomaly may be a picture that is half mountain and half city street. These anomalies are challenging since they have the same local patches as the normal images. We mixed the anomalies with normal test images and asked the detectors to find them. Some examples are shown in Fig. 4(a). The images are represented as in [15]: we treat each of them as a group of local points. On each image we randomly sample 100 patches, on each patch extract the 128-dimensional SIFT feature [20], and then reduce its dimension to 2 using PCA. Points near the stitching boundaries are discarded to avoid boundary artifacts.

We compare FGM with several other methods. We implemented a simple detector based on Gaussian mixture models (GMM); it is able to detect point-based anomalies. This method fits a GMM to all data points, calculates the points' scores as their likelihood under this GMM, and finally scores a group by averaging these numbers. To be able to detect distribution-based anomalies, we also implemented two other competitors. The first one, called LDA-KNN, uses LDA to estimate the topic distributions of the groups and treats these topic distributions (vector parameters of multinomials) as the groups' features. Then, a $k$-nearest neighbor (KNN) based point detector [21] is used to score the groups' features. The second method uses symmetrized Kullback-Leibler (KL) divergences between densities (DD). For each group, DD uses a GMM to estimate the distribution of its points. Then KL divergences between these GMMs are estimated using Monte Carlo method, and then the KNN-based detector is used to find anomalous GMMs (*i.e.*, groups).

For all algorithms we used $K = 8$ topics and $T = 6$ genres as it was suggested by BIC searches. We set $\kappa_0 = \nu_0 = 200$ for FGM. The performance is measured by the *area under the ROC curve* (AUC) of retrieving the anomalies from the test set. In the supplementary material we also show results using the *average precision* performance measure. The performances from 30 random runs are shown in Figure 4(b). GMM cannot detect the group anomalies that do not have anomalous points. The performance of LDA-KNN was also close to the $50\%$ random baseline. A possible reason is that the KNN detector did not perform well in the $K = 8$ dimensional space. MGMM, ThM, and FGM show improvements over the random baseline, and FGM achieves significantly better results than others: the *paired t-test* gives a p-value of $1.6 \times 10^{-5}$ for FGM vs. ThM. We can also see that the DD method performs poorly possibly due to many error-prone steps including fitting the GMMs and estimating divergences using Monte Carlo method.

## 6.3   Turbulence Data

We present an explorative study of detecting group anomalies on turbulence data from the JHU Turbulence Database Cluster[2] (TDC) [22]. TDC simulates fluid motion through time on a 3-dimensional grid, and here we perform our experiment on a continuous $128^3$ sub-grid. In each time step and each

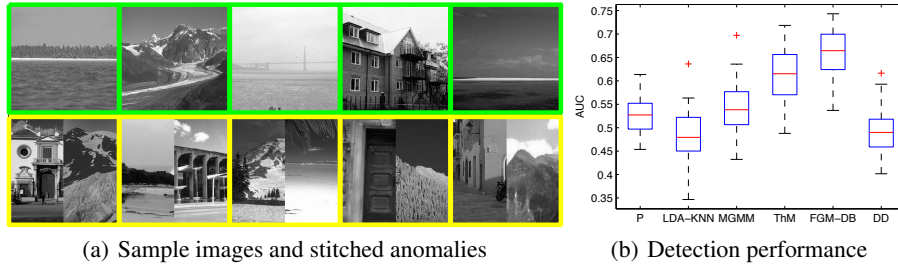

(a) Sample images and stitched anomalies          (b) Detection performance

Figure 4: Detection of stitched images. (a) Images samples. Green boxes (first row) contain natural images, and yellow boxes (second row) contain stitched anomalies. (b) The detection AUCs.

vertex of the grid, TDC records the 3-dimensional velocity of the fluid. We consider the vertices in a local cubic region as a group, and the goal is to find groups of vertices whose velocity distributions (*i.e.* moving patterns) are unusual and potentially interesting. The following steps were used to extract the groups: (1) We chose the $\{(8i, 8j, 8k)\}_{i,j,k}$ grid points as centers of our groups. Around these centers, the points in $7^3$ sized cubes formed our groups. (2) The feature of a point in the cube was its velocity relative to the velocity at its cube's center point. After these pre-processing steps, we had $M = 4\,096$ groups, each of which had $342$ 3-dimensional feature vectors.

We applied MGMM, ThM, and FGM to find anomalies in this group data. $T = 4$ genres and $K = 6$ topics were used for all methods. We do not have a groundtruth for anomalies in this data set. However, we can compute the "vorticity score" [23] for each vertex that indicates the tendency of the fluid to "spin". Vortices and especially their interactions are uncommon and of great interest in the field of fluid dynamics. This vorticity can be considered as a hand crafted anomaly score based on expert knowledge of this fluid data. We do not want an anomaly detector to match this score perfectly because there are other "non-vortex" anomalous events it should find as well. However, we do think higher correlation with this score indicates better anomaly detection performance.

Figure 5 visualizes the anomaly scores of FGM and the vorticity. We can see that these pictures are highly correlated, which implies that FGM was able to find interesting turbulence activities based on velocity only and without using the definition of vorticity or any other expert knowledge. Correlation values between vorticity and the MGMM, ThM, and FGM scores from 20 random runs are displayed in Fig. 5(c), showing that FGM is better at finding regions with high vorticity.

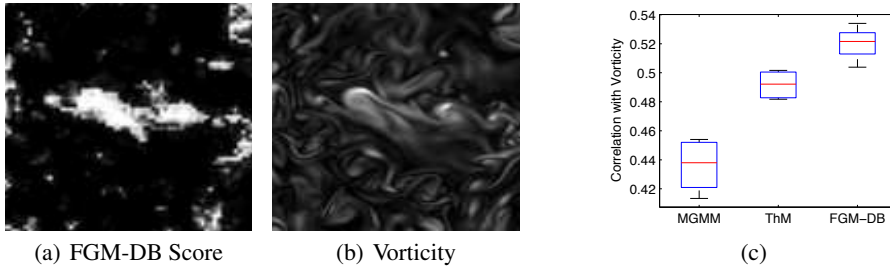

(a) FGM-DB Score          (b) Vorticity          (c)

Figure 5: Detection results for the turbulence data. (a) & (b) FGM-DB anomaly score and vorticity visualized on one slice of the cube. (c) Correlations of the anomaly scores with the vorticity.

## 7    Conclusion

We presented the generative *Flexible Genre Model* (FGM) for the group anomaly detection problem. Compared to traditional topic models, FGM is able to characterize groups' behaviors at multiple levels. This detailed characterization makes FGM an ideal tool for detecting different types of group anomalies. Empirical results show that FGM achieves better performance than existing approaches.

In the future, we will examine other possibilities as well. For model selection, we can extend FGM by using nonparametric Bayesian techniques such as hierarchical Dirichlet processes [24]. It would also be interesting to study structured groups in which the *exchangeability* assumption is not valid.

## Footnotes

[1]For example, the *Sloan Digital Sky Survey* (SDSS), http://www.sdss.org

[2]http://turbulence.pha.jhu.edu

# References

[1] Varun Chandola, Arindam Banerjee, and Vipin Kumar. Anomaly detection: A survey. *ACM Computing Surveys*, 41-3, 2009.

[2] Geoffrey G. Hazel. Multivariate gaussian MRF for multispectral scene segmentation and anomaly detection. *IEEE Trans. Geoscience and Remote Sensing*, 38-3:1199 – 1211, 2000.

[3] Kaustav Das, Jeff Schneider, and Daniel Neill. Anomaly pattern detection in categorical datasets. In *Knowledge Discovery and Data Mining (KDD)*, 2008.

[4] Kaustav Das, Jeff Schneider, and Daniel Neill. Detecting anomalous groups in categorical datasets. Technical Report 09-104, CMU-ML, 2009.

[5] Philip K. Chan and Matthew V. Mahoney. Modeling multiple time series for anomaly detection. In *IEEE International Conference on Data Mining*, 2005.

[6] Eamonn Keogh, Jessica Lin, and Ada Fu. Hot sax: Efficiently finding the most unusual time series subsequence. In *IEEE International Conference on Data Mining*, 2005.

[7] G. Mark Voit. Tracing cosmic evolution with clusters of galaxies. *Reviews of Modern Physics*, 77(1):207 – 258, 2005.

[8] B. de Finetti. Funzione caratteristica di un fenomeno aleatorio. *Atti della R. Academia Nazionale dei Lincei, Serie 6. Memorie, Classe di Scienze Fisiche, Mathematice e Naturale*, 4, 1931.

[9] Thomas Hofmann. Unsupervised learning with probabilistic latent semantic analysis. *Machine Learning Journal*, 2001.

[10] David M. Blei, Andrew Y. Ng, and Michael I. Jordan. Latent Dirichlet allocation. *JMLR*, 3:993–1022, 2003.

[11] Stuart Geman and Donald Geman. Stochastic relaxation, gibbs distributions, and the bayesian restoration of images. *IEEE Trans. PAMI*, 6:721 – 741, 1984.

[12] Gilles Celeux, Didier Chaveau, and Jean Diebolt. Stochastic version of the em algorithm: An experimental study in the mixture case. *J. of Statistical Computation and Simulation*, 55, 1996.

[13] Liang Xiong, Barnabás Póczos, and Jeff Schneider. Hierarchical probabilistic models for group anomaly detection. In *International conference on Artificial Intelligence and Statistics (AISTATS)*, 2011.

[14] Mikaela Keller and Samy Bengio. Theme-topic mixture model for document representation. In *Learning Methods for Text Understanding and Mining*, 2004.

[15] Li Fei-Fei and P. Perona. A bayesian hierarchical model for learning natural scene categories. *IEEE Conf. CVPR*, pages 524–531, 2005.

[16] Gabriel Doyle and Charles Elkan. Accounting for burstiness in topic models. In *International Conference on Machine Learning*, 2009.

[17] Andrew Gelman, John B. Carlin, Hal S. Stern, and Donald B. Rubin. *Bayesian Data Analysis*. Chapman and Hall/CRC, 2003.

[18] Thomas P. Minka. Estimating a dirichlet distribution. `http://research.microsoft.com/en-us/um/people/minka/papers/dirichlet`, 2009.

[19] Gideon E. Schwarz. Estimating the dimension of a model. *Annals of Statistics*, (6-2):461–464, 1974.

[20] David G. Lowe. Distinctive image features from scale-invariant keypoints. *IJCV*, 60(2):91 – 110, 2004.

[21] Manqi Zhao. Anomaly detection with score functions based on nearest neighbor graphs. In *NIPS*, 2009.

[22] E. Perlman, R. Burns, Y. Li, and C. Meneveau. Data exploration of turbulence simulations using a database cluster. In *Supercomputing SC*, 2007.

[23] Charles Meneveau. Lagrangian dynamics and models of the velocity gradient tensor in turbulent flows. *Annual Review of Fluid Mechanics*, 43:219–45, 2011.

[24] Yee Whye Teh, Michael I. Jordan, Matthew J. Beal, and David M. Blei. Hierarchical Dirichlet process. *Journal of the American Statistical Association*, 101:1566 – 1581, 2006.

